# Viewpoint invariant face recognition using independent component analysis and attractor networks

**Marian Stewart Bartlett**
University of California San Diego
The Salk Institute
La Jolla, CA 92037
marni@salk.edu

**Terrence J. Sejnowski**
University of California San Diego
Howard Hughes Medical Institute
The Salk Institute, La Jolla, CA 92037
terry@salk.edu

## Abstract

We have explored two approaches to recognizing faces across changes in pose. First, we developed a representation of face images based on independent component analysis (ICA) and compared it to a principal component analysis (PCA) representation for face recognition. The ICA basis vectors for this data set were more spatially local than the PCA basis vectors and the ICA representation had greater invariance to changes in pose. Second, we present a model for the development of viewpoint invariant responses to faces from visual experience in a biological system. The temporal continuity of natural visual experience was incorporated into an attractor network model by Hebbian learning following a lowpass temporal filter on unit activities. When combined with the temporal filter, a basic Hebbian update rule became a generalization of Griniasty et al. (1993), which associates temporally proximal input patterns into basins of attraction. The system acquired representations of faces that were largely independent of pose.

## 1 Independent component representations of faces

Important advances in face recognition have employed forms of principal component analysis, which considers only second-order moments of the input (Cottrell & Metcalfe, 1991; Turk & Pentland 1991). Independent component analysis (ICA) is a generalization of principal component analysis (PCA), which decorrelates the higher-order moments of the input (Comon, 1994). In a task such as face recognition, much of the important information is contained in the high-order statistics of the images. A representational basis in which the high-order statistics are decorrelated may be more powerful for face recognition than one in which only the second order statistics are decorrelated, as in PCA representations. We compared an ICA-based representation to a PCA-based representation for recognizing faces across changes in pose.

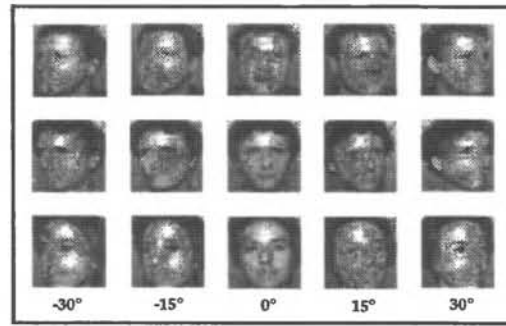

Figure 1: Examples from image set (Beymer, 1994).

The image set contained 200 images of faces, consisting of 40 subjects at each of five poses (Figure 1). The images were converted to vectors and comprised the rows of a 200 x 3600 data matrix, $X$. We consider the face images in $X$ to be a linear mixture of an unknown set of statistically independent source images $S$, where $A$ is an unknown mixing matrix (Figure 2). The sources are recovered by a matrix of learned filters, $W$, which produce statistically independent outputs, $U$.

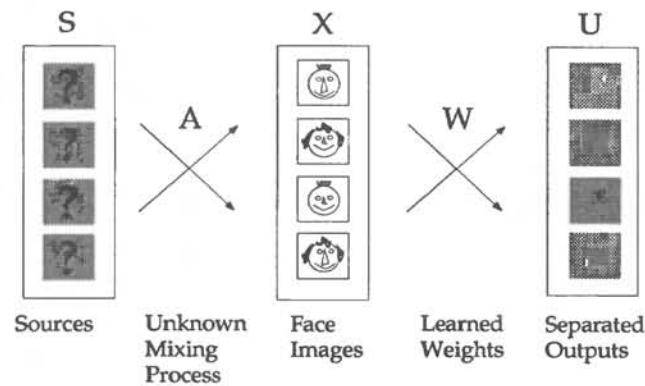

Figure 2: Image synthesis model.

The weight matrix, $W$, was found through an unsupervised learning algorithm that maximizes the mutual information between the input and the output of a nonlinear transformation (Bell & Sejnowski, 1995). This algorithm has proven successful for separating randomly mixed auditory signals (the cocktail party problem), and has recently been applied to EEG signals (Makeig et al., 1996) and natural scenes (see Bell & Sejnowski, this volume). The independent component images contained in the rows of $U$ are shown in Figure 3. In contrast to the principal components, all 200 independent components were spatially local. We took as our face representation the rows of the matrix $A = W^{-1}$ which provide the linear combination of source images in $U$ that comprise each face image in $X$.

## 1.1   Face Recognition Performance: ICA vs. Eigenfaces

We compared the performance of the ICA representation to that of the PCA representation for recognizing faces across changes in pose. The PCA representation of a face consisted of its component coefficients, which was equivalent to the "Eigenface"

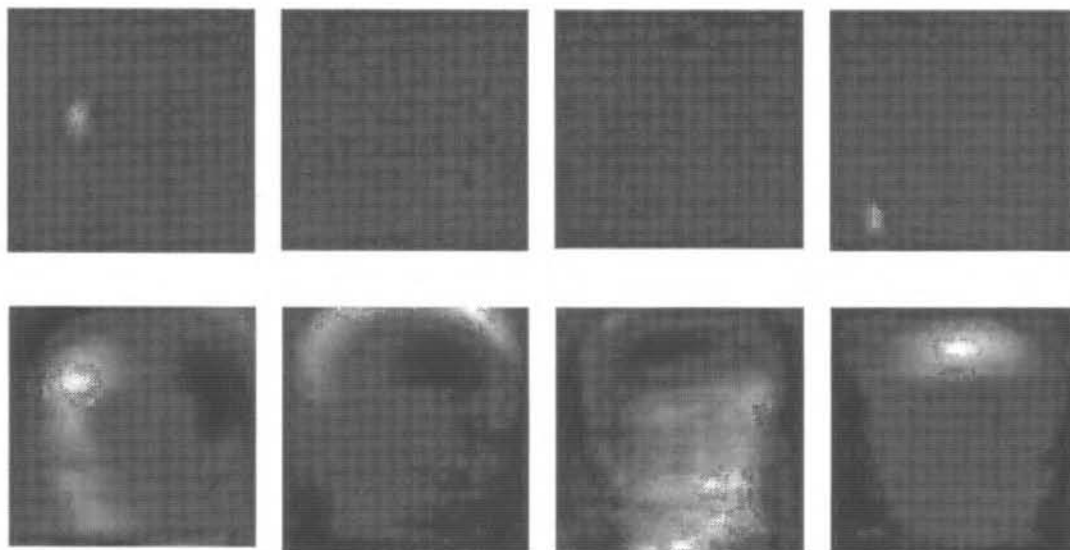

Figure 3: Top: Four independent components of the image set. Bottom: First four principal components.

representation (Turk & Pentland, 1991). A test image was recognized by assigning it the label of the nearest of the other 199 images in Euclidean distance.

Classification error rates for the ICA and PCA representations and for the original graylevel images are presented in Table 1. For the PCA representation, the best performance was obtained with the 120 principal components corresponding to the highest eigenvalues. Dropping the first three principal components, or selecting ranges of intermediate components did not improve performance. The independent component sources were ordered by the magnitude of the weight vector, row of $W$, used to extract the source from the image.[1] Best performance was obtained with the 130 independent components with the largest weight vectors. Performance with the ICA representation was significantly superior to Eigenfaces by a paired t-test ($\alpha < 0.05$).

|  | Mutual Information | Percent Correct Recognition |
|---|---|---|
| Graylevel Images | .89 | .83 |
| PCA | .10 | .84 |
| ICA | .007 | .87 |

Table 1: Mean mutual information between all pairs of 10 basis images, and between the original graylevel images. Face recognition performance is across all 200 images.

For the task of recognizing faces across pose, a statistically independent basis set provided a more powerful representation for face images than a principal component representation in which only the second order statistics are decorrelated.

## 2  Unsupervised Learning of Viewpoint Invariant Representations of Faces in an Attractor Network

Cells in the primate inferior temporal lobe have been reported that respond selectively to faces despite substantial changes in viewpoint (Hasselmo, Rolls, Baylis, & Nalwa, 1989). Some cells responded independently of viewing angle, whereas other cells gave intermediate responses between a viewer-centered and an object centered representation. This section addresses how a system can acquire such invariance to viewpoint from visual experience.

During natural visual experience, different views of an object or face tend to appear in close temporal proximity as an animal manipulates the object or navigates around it, or as a face changes pose. Capturing such temporal relationships in the input is a way to automatically associate different views of an object without requiring three dimensional descriptions.

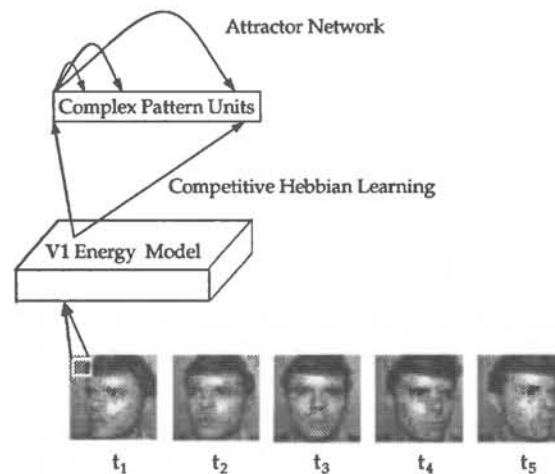

Figure 4: Model architecture.

Hebbian learning can capture these temporal relationships in a feedforward system when the output unit activities are passed through a lowpass temporal filter (Foldiak, 1991; Wallis & Rolls, 1996). Such lowpass temporal filters have been related to the time course of the modifiable state of a neuron based on the open time of the NMDA channel for calcium influx (Rhodes, 1992). We show that 1) this lowpass temporal filter increases viewpoint invariance of face representations in a feedforward system trained with competitive Hebbian learning, and 2) when the input patterns to an attractor network are passed through a lowpass temporal filter, then a basic Hebbian weight update rule associates sequentially proximal input patterns into the same basin of attraction.

This simulation used a subset of 100 images from Section 1, consisting of twenty faces at five poses each. Images were presented to the model in sequential order as the subject changed pose from left to right (Figure 4). The first layer is an energy model related to the output of V1 complex cells (Heeger, 1991). The images were filtered by a set of sine and cosine Gabor filters at 4 spatial scales and 4 orientations at 255 spatial locations. Sine and cosine outputs were squared and summed. The set of V1 model outputs projected to a second layer of 70 units, grouped into two

inhibitory pools. The third stage of the model was an attractor network produced by lateral interconnections among all of the complex pattern units. The feedforward and lateral connections were trained successively.

## 2.1 Competitive Hebbian learning of temporal relationships

The Competitive Learning Algorithm (Rumelhart & Zipser, 1985) was extended to include a temporal lowpass filter on output unit activities (Bartlett & Sejnowski, 1996). This manipulation gives the winner in the previous time steps a competitive advantage for winning, and therefore learning, in the current time step.

$$\Delta w_{ij} = \begin{cases} \alpha(\frac{x_{iu}}{s_u} - w_{ij}) & \text{if } winner = j \\ 0.1\alpha(\frac{x_{iu}}{s_u} - w_{ij}) & \text{if } winner \neq j \end{cases}$$

$$\begin{aligned} winner &= max_j[\overline{y_j}^{(t)}] \\ \overline{y_j}^{(t)} &= \lambda y_j^t + (1 - \lambda)\overline{y_j}^{(t-1)} \end{aligned} \tag{1}$$

The output activity of unit $j$ at time $t$, $\overline{y_j}^{(t)}$, is determined by the trace, or running average, of its activation, where $y_j^t$ is the weighted sum of the feedforward inputs, $\alpha$ is the learning rate, $x_{iu}$ is the value of input unit $i$ for pattern $u$, and $s_u$ is the total amount of input activation for pattern $u$. The weight to each unit was constrained to sum to one. This algorithm was used to train the feedforward connections. There was one face pattern per time step and $\lambda$ was set to 1 between individuals.

## 2.2 Lateral connections in the output layer form an attractor network

Hebbian learning of lateral interconnections, in combination with a lowpass temporal filter on the unit activities in (1), produces a learning rule that associates temporally proximal inputs into basins of attraction. We begin with a simple Hebbian learning rule

$$W_{ij} = \frac{1}{N} \sum_{t=1}^{P} (y_i^t - y^0)(y_j^t - y^0) \tag{2}$$

where $N$ is the number of units, $P$ is the number of patterns, and $y^0$ is the mean activity over all of the units. Replacing $y_i^t$ with the activity trace $\overline{y_i}^{(t)}$ defined in (1), substituting $y^0 = \lambda y^0 + (1 - \lambda)y^0$ and multiplying out the terms leads to the following learning rule:

$$W_{ij} = \frac{1}{N} \sum_{t=1}^{P}(y_i^t - y^0)(y_j^t - y^0) + k_1 \left[ (y_i^t - y^0)(\overline{y_j}^{(t-1)} - y^0) + (\overline{y_i}^{(t-1)} - y^0)(y_j^t - y^0) \right]$$

$$+ k_2 \left[ (\overline{y_i}^{(t-1)} - y^0)(\overline{y_j}^{(t-1)} - y^0) \right] \tag{3}$$

$$\text{where } k_1 = \frac{\lambda(1-\lambda)}{\lambda^2} \text{ and } k_2 = \frac{(1-\lambda)^2}{\lambda^2}$$

The first term in this equation is basic Hebbian learning, the second term associates pattern $t$ with pattern $t - 1$, and the third term is Hebbian association of the trace activity for pattern $t - 1$. This learning rule is a generalization of an attractor network learning rule that has been shown to associate random input patterns

into basins of attraction based on serial position in the input sequence (Griniasty, Tsodyks & Amit, 1993). The following update rule was used for the activation $V$ of unit $i$ at time $t$ from the lateral inputs (Griniasty, Tsodyks, & Amit, 1993):

$$V_i(t + \delta t) = \phi \left[ \sum W_{ij} V_j(t) - \theta \right]$$

Where $\theta$ is a neural threshold and $\phi(x) = 1$ for $x > 0$, and 0 otherwise. In these simulations, $\theta = 0.007$, $N = 70$, $P = 100$, $y^0 = 0.03$, and $\lambda = 0.5$ gave $k_1 = k_2 = 1$.

## 2.3   Results

Temporal association in the feedforward connections broadened the pose tuning of the output units (Figure 5 Left). When the lateral connections in the output layer were added, the attractor network acquired responses that were largely invariant to pose (Figure 5 Right).

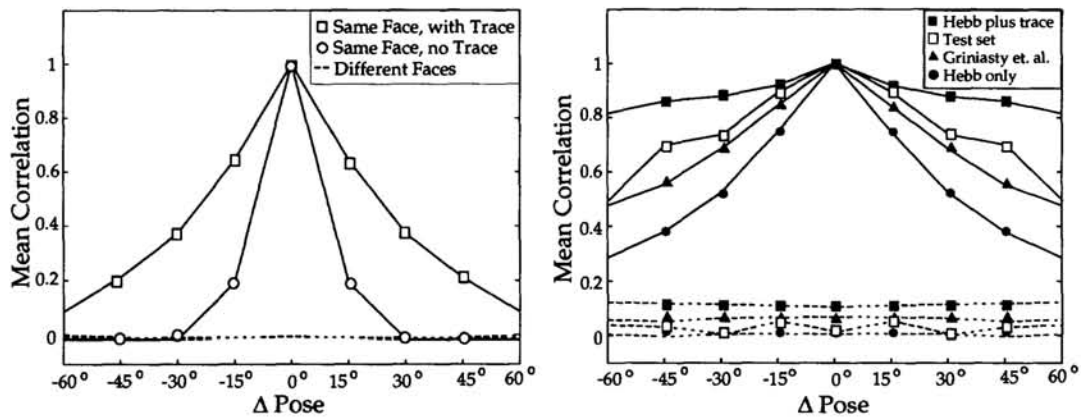

Figure 5:    Left: Correlation of the outputs of the feedforward system as a function of change in pose. Correlations across different views of the same face (−) are compared to correlations across different faces (−−) with the temporal trace parameter $\lambda = 0.5$ and $\lambda = 0$. Right: Correlations in sustained activity patterns in the attractor network as a function of change in pose. Results obtained with Equation 3 (Hebb plus trace) are compared to Hebb only, and to the learning rule in Griniasty et al. (1993). Test set results for Equation 3 (open squares) were obtained by alternately training on four poses and testing on the fifth, and then averaging across all test cases.

| F | N | Attractor Network | | ICA |
|---|---|---|---|---|
| | | F/N | % Correct | % Correct |
| 5 | 70 | .07 | 1.00 | .96 |
| 10 | 70 | .14 | .90 | .86 |
| 20 | 70 | .29 | .61 | .89 |
| 20 | 160 | .13 | .73 | .89 |

Table 2: Face classification performance of the attractor network for four ratios of the number of desired memories, F, to the number of units, N. Results are compared to ICA for the same subset of faces.

Classification accuracy of the attractor network was calculated by nearest neighbor on the activity states (Table 2). Performance of the attractor network depends both on the performance of the feedforward system, which comprises its input, and on the ratio of the number of patterns to be encoded in memory, F, to the number of units, N, where each individual in the face set comprises one memory pattern. The attractor network performed well when this ratio was sufficiently high. The ICA representation also performed well, especially for N=20.

The goal of this simulation was to begin with structured inputs similar to the responses of V1 complex cells, and to explore the performance of unsupervised learning mechanisms that can transform these inputs into pose invariant responses. We showed that a lowpass temporal filter on unit activities, which has been related to the time course of the modifiable state of a neuron (Rhodes, 1992), cooperates with Hebbian learning to (1) increase the viewpoint invariance of responses to faces in a feedforward system, and (2) create basins of attraction in an attractor network which associate temporally proximal inputs. These simulations demonstrated that viewpoint invariant representations of complex objects such as faces can be developed from visual experience by accessing the temporal structure of the input.

## Acknowledgments
This project was supported by Lawrence Livermore National Laboratory ISCR Agreement B291528, and by the McDonnell-Pew Center for Cognitive Neuroscience at San Diego.

## Footnotes

[1]The magnitude of the weight vector for optimally projecting the source onto the sloping part of the nonlinearity provides a measure of the variance of the original source (Tony Bell, personal communication).

## References
Bartlett, M. Stewart, & Sejnowski, T., 1996. Unsupervised learning of invariant representations of faces through temporal association. *Computational Neuroscience: Int. Rev. Neurobio. Suppl. 1* J.M Bower, Ed., Academic Press, San Diego, CA:317-322.

Beymer, D. 1994. Face recognition under varying pose. In *Proceedings of the 1994 IEEE Computer Society Conference on Computer Vision and Pattern Recognition.* Los Alamitos, CA: IEEE Comput. Soc. Press: 756-61.

Bell, A. & Sejnowski, T., (1997). The independent components of natural scenes are edge filters. *Advances in Neural Information Processing Systems 9.*

Bell, A., & Sejnowski, T., 1995. An information Maximization approach to blind separation and blind deconvolution. *Neural Comp.* 7: 1129-1159.

Comon, P. 1994. Independent component analysis - a new concept? *Signal Processing* 36:287-314.

Cottrell & Metcalfe, 1991. Face, gender and emotion recognition using Holons. In *Advances in Neural Information Processing Systems 3*, D. Touretzky, (Ed.), Morgan Kaufman, San Mateo, CA: 564 - 571.

Foldiak, P. 1991. Learning invariance from transformation sequences. *Neural Comp.* 3:194-200.

Griniasty, M., Tsodyks, M., & Amit, D. 1993. Conversion of temporal correlations between stimuli to spatial correlations between attractors. *Neural Comp.* 5:1-17.

Hasselmo M. Rolls E. Baylis G. & Nalwa V. 1989. Object-centered encoding by face-selective neurons in the cortex in the superior temporal sulcus of the monkey. *Experimental Brain Research* 75(2):417-29.

Heeger, D. (1991). Nonlinear model of neural responses in cat visual cortex. *Computational Models of Visual Processing*, M. Landy & J. Movshon, Eds. MIT Press, Cambridge, MA.

Makeig, S, Bell, AJ, Jung, T-P, and Sejnowski, TJ 1996. Independent component analysis of Electroencephalographic data, In: *Advances in Neural Information Processing Systems* 8, 145-151.

Rhodes, P. 1992. The long open time of the NMDA channel facilitates the self-organization of invariant object responses in cortex. *Soc. Neurosci. Abst.* 18:740.

Rumelhart, D. & Zipser, D. 1985. Feature discovery by competitive learning. *Cognitive Science* 9: 75-112.

Turk, M., & Pentland, A. 1991. Eigenfaces for Recognition. *J. Cog. Neurosci.* 3(1):71-86.

Wallis, G. & Rolls, E. 1996. A model of invariant object recognition in the visual system. Technical Report, Oxford University Department of Experimental Psychology.